# The Unified Propagation and Scaling Algorithm

**Yee Whye Teh**
Department of Computer Science
University of Toronto
10 King's College Road
Toronto  M5S 3G4  Canada
*ywteh@cs.toronto.edu*

**Max Welling**
Gatsby Computational Neuroscience Unit
University College London
17 Queen Square
London  WC1N 3AR  U.K.
*welling@gatsby.ucl.ac.uk*

## Abstract

In this paper we will show that a restricted class of constrained minimum divergence problems, named generalized inference problems, can be solved by approximating the KL divergence with a Bethe free energy. The algorithm we derive is closely related to both loopy belief propagation and iterative scaling. This *unified propagation and scaling* algorithm reduces to a convergent alternative to loopy belief propagation when no constraints are present. Experiments show the viability of our algorithm.

## 1   Introduction

For many interesting models, exact inference is intractible. Trees are a notable exception where Belief Propagation (BP) can be employed to compute the posterior distribution [1]. BP on loopy graphs can still be understood as a form of approximate inference since its fixed points are stationary points of the Bethe free energy [2]. A seemingly unrelated problem is that of finding the distribution with minimim KL divergence to a prior distribution subject to some constraints. This problem can be solved through the iterative scaling (IS) procedure [3]. Although a lot of work has been done on approximate inference, there seems to be no counterpart in the literature on *approximate* minimum divergence problems. This paper shows that the Bethe free energy can be used as an approximation to the KL divergence and derives a novel approximate minimum divergence algorithm which we call *unified propagation and scaling* (UPS).

In section 2 we introduce generalized inference and the iterative scaling (IS) algorithm. In section 3, we approximate the KL divergence with the Bethe free energy and derive fixed point equations to perform approximate generalized inference. We also show in what sense our fixed point equations are related to loopy BP and IS. Section 4 describes unified propagation and scaling (UPS), a novel algorithm to minimize the Bethe free energy, while section 5 shows experiments on the efficiency and accuracy of UPS.

## 2   Generalized Inference

In this section we will introduce generalized inference and review some of the literature on iterative scaling (IS). Let $x = \{x_i\}$ where $x_i$ is the variable associated with node $i$. Consider an undirected graphical model $G$ with single and pairwise potentials $\psi_i(x_i)$,

$\psi_{ij}(x_i, x_j)$. Let $P(x)$ be the distribution represented by $G$, i.e.

$$P(x) \propto \prod_{(ij)} \psi_{ij}(x_i, x_j) \prod_i \psi_i(x_i) \propto \prod_{(ij)} \phi_{ij}(x_i, x_j) \prod_i \phi_i(x_i)^{1-n_i} \tag{1}$$

where $\phi_{ij}(x_i, x_j) = \psi_{ij}(x_i, x_j)\psi_i(x_i)\psi_j(x_j)$, $\phi_i(x_i) = \psi_i(x_i)$, $(ij)$ ranges over the edges of $G$, $i$ ranges over the nodes of $G$ and $n_i$ is the number of neighbours of $i$. Let $V$ be a subset of nodes. For $i \in V$ let $\widehat{o}_i(x_i)$ be a fixed distribution over $x_i$. Given these "observed distributions" on $V$, define the generalized posterior as the distribution $Q(x)$ which minimizes the KL divergence

$$KL(Q\|P) = \sum_x Q(x)\left(\log Q(x) - \log P(x)\right) \tag{2}$$

subject to the constraints that $Q(x_i) = \widehat{o}_i(x_i)$ for each $i \in V$. We call these constraints observational (Obs) constraints. Generalized inference is the process by which we determine the generalized posterior[1]. Let $H$ be the set of unobserved nodes, i.e. all nodes not in $V$.

**Theorem 1** *If* $\widehat{o}_i(x_i) = \delta(x_i - \widehat{x}_i)$ *for each* $i \in V$ *then the generalized posterior is* $Q(x) = P\left(x_H \mid \widehat{x}_V\right) \prod_{i \in V} \delta(x_i - \widehat{x}_i)$.

where $x_S = \{x_i\}_{i \in S}$ for a subset of nodes $S$. Similarly if $S$ is a subgraph of $G$. The above theorem shows that if the constrained marginals are delta functions, i.e. the observations are hard, then the generalized posterior reduces to a trivial extension of the ordinary posterior, hence explaining our use of the term generalized inference.

Since generalized inference is a constrained minimum divergence problem, a standard way of solving it is using Lagrange multipliers. For each $i \in V$ and $x_i$, let $\lambda_i(x_i)$ be the Lagrange multiplier enforcing $Q(x_i) = \widehat{o}_i(x_i)$. Then the generalized posterior is

$$Q(x) \propto P(x) \prod_{i \in V} e^{\lambda_i(x_i)} \propto \prod_{(ij)} \psi_{ij}(x_i, x_j) \prod_{i \in H} \psi_i(x_i) \prod_{i \in V} \psi_i(x_i)e^{\lambda_i(x_i)} \tag{3}$$

where we chose $\lambda_i(x_i)$ to satisfy the Obs constraints. Iterative scaling (IS) can now be used to solve for $\lambda_i(x_i)$ [3]. At each iteration of IS, the Lagrange multiplier $\lambda_i(x_i)$ is updated using the IS scaling update

$$e^{\lambda_i(x_i)} \leftarrow e^{\lambda_i(x_i)} \frac{\widehat{o}_i(x_i)}{Q(x_i)} \qquad \text{for each } x_i \tag{4}$$

Intuitively, (4) updates the current posterior so that the marginal $Q(x_i)$ for node $i$ match the given constraint $\widehat{o}_i(x_i)$. IS is a specific case of the generalized iterative scaling (GIS) algorithm [4], which updates the Lagrange multipliers for a subset $U \subset V$ of nodes using $e^{\lambda_i(x_i)} \leftarrow e^{\lambda_i(x_i)}(\frac{\widehat{o}_i(x_i)}{Q(x_i)})^{1/|U|}$. Parallel GIS steps can be understood as performing IS updates in parallel, but damping the steps such that the algorithm is still guaranteed to converge.

Ordinary inference is needed to compute the current marginals $Q(x_i)$ required by (4). If $G$ is singly connected, then belief propagation (BP) can be used to compute the required marginals. Otherwise, exact inference or sampling algorithms like Markov chain Monte Carlo can be used, but usually are computationally taxing. Alternative approximate inference algorithms like variational methods and loopy BP can be used instead to estimate the

required marginals. Although being much more efficient, they can also produce biased estimates, potentially leading to the overall IS not converging[2]. Even if IS did converge, we do not have much theoretical understanding of the accuracy of the overall algorithm.

A more principled approach is to first approximate the KL divergence, then derive algorithms to minimize the approximation. In the next section, we describe a Bethe free energy approximation to the KL divergence. Fixed point equations for minimizing the Bethe approximation can then be derived. The fixed point equations reduce to BP propagation updates at hidden nodes, and to IS scaling updates at observed nodes. As a consequence, using loopy BP to approximate the required marginals turns out to be a particular scheduling of the fixed point equations. Because the Bethe free energy is fairly well understood, and is quite accurate in many regimes [5, 2, 6], we conclude that IS with loopy BP is a viable approximate generalized inference technique. However, in section 4 we describe more efficient algorithms for approximate generalized inference based upon the Bethe free energy.

## 3    Approximate Generalized Inference

Let $b_{ij}(x_i, x_j)$ and $b_i(x_i)$ be estimates of the pair-wise and single site marginals of the generalized posterior. $b_{ij}(x_i, x_j)$ and $b_i(x_i)$ are called beliefs. The beliefs need to satisfy the following marginalization and normalization (MN) constraints:

$$\sum_{x_j} b_{ij}(x_i, x_j) = b_i(x_i) \qquad\qquad \sum_{x_i} b_i(x_i) = 1 \qquad\qquad (5)$$

Let $Q = \{b_{ij}(x_i, x_j), b_i(x_i)\}$. The Bethe free energy is defined as

$$\mathcal{F}_{bethe}(Q) = \sum_{(ij)} b_{ij}(x_i, x_j) \log \frac{b_{ij}(x_i, x_j)}{\phi_{ij}(x_i, x_j)} + \sum_i (1 - n_i) b_i(x_i) \log \frac{b_i(x_i)}{\phi_i(x_i)} \qquad (6)$$

$\mathcal{F}_{bethe}$ is an approximation to the KL divergence which only accounts for pair-wise correlations between neighbouring variables and is exact if $G$ is singly connected.

We wish to minimize $\mathcal{F}_{bethe}(Q)$ subject to the MN and Obs constraints. We use Lagrange multipliers $\lambda_{ji}(x_i)$ to impose the marginalization constraints. We can also use Lagrange multipliers to impose the normalization and observational constraints as well, but this reduces to simply keeping $b_{ij}(x_i, x_j)$ and $b_i(x_i)$ normalized, and keeping $b_i(x_i) = \widehat{o}_i(x_i)$ fixed for $i \in V$. We shall ignore these for clarity. The resulting Lagrangian is

$$L = \mathcal{F}_{bethe}(Q) - \sum_i \sum_{j \in N(i)} \lambda_{ji}(x_i) \left( \sum_{x_j} b_{ij}(x_i, x_j) - b_i(x_i) \right) \qquad (7)$$

where $N(i)$ denotes the set of neighbours of node $i$. Setting derivatives of $L$ with respect to $b_{ij}(x_i, x_j)$, $b_i(x_i)$ and $\lambda_{ji}(x_i)$ to 0, we get

**Theorem 2** *Subject to the MN and Obs constraints, every stationary point of $\mathcal{F}_{bethe}$ is given by*

$$b_{ij}(x_i, x_j) \propto \phi_{ij}(x_i, x_j) e^{\lambda_{ji}(x_i) + \lambda_{ij}(x_j)} \qquad b_i(x_i) \propto \phi_i(x_i) e^{\frac{1}{n_i - 1} \sum_{j \in N(i)} \lambda_{ji}(x_i)} \qquad (8)$$

*where the Lagrange multipliers are fixed points of the following updates:*

$$e^{\lambda_{ji}(x_i)} \leftarrow \prod_{k \in N(i) \backslash j} \sum_{x_k} \frac{\phi_{ik}(x_i, x_k)}{\phi_i(x_i)} e^{\lambda_{ik}(x_k)} \qquad \text{for } i \notin V, j \in N(i) \qquad (9)$$

$$e^{\lambda_{ji}(x_i)} \leftarrow \frac{\widehat{o}_i(x_i)}{\sum_{x_j} \phi_{ij}(x_i, x_j) e^{\lambda_{ij}(x_j)}} \qquad \text{for } i \in V, j \in N(i) \qquad (10)$$

Equation (9) is equivalent to the BP propagation updates by identifying the messages as $M_{ij}(x_j) = \sum_{x_i} \frac{\phi_{ij}(x_i,x_j)}{\phi_j(x_j)} e^{\lambda_{ji}(x_i)}$ [3]. Rewriting (10) in terms of messages as well we find,

$$M_{ij}(x_j) \leftarrow \sum_{x_i} \psi_{ij}(x_i, x_j) \frac{\widehat{o}_i(x_i)}{M_{ji}(x_i)} \qquad \text{for } i \in V, j \in N(i) \qquad (11)$$

We can extend the analogy and understand (11) as a message "bouncing" step, in which messages going into an observed node get bounced back and are altered in the process. If $\widehat{o}_i(x_i) = \delta(x_i, \widehat{x}_i)$ is a delta function, then (11) reduces to $M_{ij}(x_j) \leftarrow \psi_{ij}(\widehat{x}_i, x_j)$ so that instead of bouncing back, messages going into node $i$ get absorbed. An alternative description of (10) is given by the following theorem.

**Theorem 3** *Let $i \in V$. Updating each $\lambda_{ji}(x_i)$ for $j \in N(i)$ using (10) is equivalent to updating $\lambda_i(x_i)$ using (4), where we identify*

$$Q(x_i) \propto \phi_i(x_i) \prod_{j \in N(i)} M_{ji}(x_i) \qquad e^{\lambda_i(x_i)} = \left( \frac{\phi_i(x_i)}{\widehat{o}_i(x_i)} \right)^{n_i - 1} \prod_{j \in N(i)} e^{\lambda_{ji}(x_i)} \qquad (12)$$

Theorem 3 states the unexpected result that scaling updates (4) are just fixed point equations to minimize $\mathcal{F}_{bethe}$. Further, the required marginals $Q(x_i)$ are computed using (9), which is exactly loopy BP. Hence using loopy BP to approximate the marginals required by IS is just a particular scheduling of the fixed point equations (9,10).

## 4 Algorithms to Minimize the Bethe Free Energy

Inspired by [2], we can run run the fixed point equations (9,10) and hope that they converge to a minimum of $\mathcal{F}_{bethe}$. We call this algorithm loopy IS. Theorem 2 states that if loopy IS converges it will converge to stationary points of $\mathcal{F}_{bethe}$. In simulations we find that it always gets to a good local minimum, if not the global minimum. However loopy IS does not necessarily converge, especially when the variables are strongly correlated. There are two reasons why it can fail to converge. Firstly, the loopy BP component (9) may fail to converge. However this is not serious as past results indicate that loopy BP often fails only when the Bethe approximation is not accurate [6]. Secondly, the IS component (10) may fail to converge, since it is not run sequentially and the estimated marginals are inaccurate. We will show in section 5 that this is a serious problem for loopy IS.

One way to mitigate the second problem is to use the scaling updates (4), and approximate the required marginals using an inner phase of loopy BP (call this algorithm IS+BP). Theorem 3 shows that IS+BP is just a particular scheduling of loopy IS, hence it inherits the accuracy of loopy IS while converging more often. However because we have to run loopy BP until convergence for each scaling update, IS+BP is not particularly efficient. Another way to promote convergence is to damp the loopy IS updates. This works well in practice. In this section, we describe yet another possibility – an efficient algorithm based on the

same fixed point equations (9,10) which is guaranteed to converge without damping. In subsection 4.1 we describe UPS-T, an algorithm which applies when $G$ is a tree and the Obs constraints are on the leaves of $G$. In subsection 4.2 we describe UPS for the general case, which will make use of UPS-T as a subroutine.

## 4.1 Constraining the leaves of trees

Suppose that $G$ is a tree, and all observed nodes $i \in V$ are leaves of $G$. Since $G$ is a tree, the Bethe free energy is exact, i.e. if the MN constraints are satisfied then $\mathcal{F}_{bethe} = KL(Q\|P)$ where $Q(x) = \prod_{(ij)} b_{ij}(x_i, x_j) \prod_i b_i(x_i)^{1-n_i}$. As a consequence, $\mathcal{F}_{bethe}$ is convex in the subspace defined by the MN constraints. Therefore if the fixed point equations (9,10) converge, they will converge to the unique global minimum. Further, since (9) is exactly a propagation update, and (10) is exactly a scaling update, the following scheduling of (9,10) will always converge: alternately run (9) until convergence and perform a single (10) update. The schedule essentially implements the IS+BP procedure, except that loopy BP is exact for a tree. Our algorithm essentially implements the scheduling, except that unnecessary propagation updates are not performed.

**Algorithm UPS-T** *Unified Propagation and Scaling on Trees*

1. Run propagation updates (9) until convergence.
2. Let $i_1, i_2, i_3, \ldots \in V$ be such that every node occurs infinitely often.
3. For $t = 1, 2, 3, \ldots$ until convergence criterion is met:
4.      Perform scaling update (10) for $\lambda_{j_t i_t}(x_{i_t})$, where $j_t$ is the unique neighbour of $i_t$.
5.      For each edge $u \to v$ on path from $i_t$ to $i_{t+1}$, apply propagation update (9) for $\lambda_{vu}(x_u)$.
6. Run propagation updates (9) until convergence.

## 4.2 Graphs with cycles

For graphs with cycles, $\mathcal{F}_{bethe}$ is not exact nor convex. However we can make use of the fact that it is exact on trees to find a local minimum (or saddle point). The idea is that we clamp a number of hidden nodes to their current marginals such that the rest of the hidden nodes become singly connected, and apply UPS-T. Once UPS-T has converged, we clamp a different set of hidden nodes and apply UPS-T again. The algorithm can be understood as coordinate descent where we minimize $\mathcal{F}_{bethe}$ with respect to the unclamped nodes at each iteration.

Let $C \subset H$ be a set of clamped nodes such that every loop in the graph $G$ contains a node from $U = V \cup C$. Define $G'$ to be the graph obtained from $G$ as follows. For each node $i \in U$ replicate it $n_i$ times, and connect each replica to one neighbour of $i$ and no other nodes. This is shown in figures 1(c) and 1(d) for the graph in 1(a). Clearly $G'$ will be singly connected. Let $T \subset G'$ denote the trees in $G'$. Define[4]

$$P'(x) = \prod_{T \subset G'} P'_T(x_T) \propto \prod_{T \subset G'} \prod_{(i'j') \in T} \psi_{i'j'}(x_{i'}, x_{j'}) \prod_{i' \in T} \psi_{i'}(x_{i'})^{1-n_{i'}} \qquad (13)$$

$$Q'(x) = \prod_{T \subset G'} Q'_T(x_T) \propto \prod_{T \subset G'} \prod_{(i'j') \in T} b_{i'j'}(x_{i'}, x_{j'}) \prod_{i' \in T} b_{i'}(x_{i'})^{1-n_{i'}} \qquad (14)$$

where $n_{i'}$ is the number of neighbours of node $i'$ in $G'$. By regrouping terms in $\mathcal{F}_{bethe}$ we can show the following:

**Theorem 4** *Let $\widehat{c}_i(x_i)$ be a distribution over $x_i$ for $i \in C$. Then in the subspace defined by $b_i(x_i) = \widehat{c}_i(x_i)$ for $i \in C$ and by the MN and Obs constraints, we have*

$$\mathcal{F}_{bethe} = \sum_{T \subset G'} KL(Q'_T \| P'_T) + \sum_{i \in U} (1 - n_i) \sum_{x_i} \widehat{c}_i(x_i) \log \frac{\widehat{c}_i(x_i)}{\phi_i(x_i)} \qquad (15)$$

To minimize $\mathcal{F}_{bethe}$, now all we have to do is to minimize each $KL(Q'_T \| P'_T)$ individually. We can already solve this using UPS-T. By clamping the marginals of nodes in $C$, we have reduced the problem to one solved by UPS-T, where the observed nodes are taken to include those in $C$. The overall algorithm is

**Algorithm UPS** *Unified Propagation and Scaling*

1. Initialize beliefs $Q^{(0)} = \{b_{ij}^{(0)}(x_i, x_j), b_i^{(0)}(x_i)\}$.

2. For $t = 1, 2, 3, \ldots$ until convergence criteria is met:

3.     Find a set of nodes $C^{(t)}$ such that every loopy is broken by $V \cup C^{(t)}$.

4.     Using UPS-T, set $Q^{(t)} = \mathrm{argmin}\{\mathcal{F}_{bethe}(Q) \mid b_i(x_i) = b_i^{(t-1)}(x_i)$ for $i \in C^{(t)}$, and MN and Obs constraints are satisfied $\}$.

It is clear that $\mathcal{F}_{bethe}(Q^{(t)}) \le \mathcal{F}_{bethe}(Q^{(t-1)})$ for all $t$. Now by using the fact that both scaling and propagation updates are fixed point equations for finding stationary points of $\mathcal{F}_{bethe}$ we have,

**Theorem 5** *If for all $t$ and $i \in H$ there is a $t_1 \ge t$ with $i \notin C^{(t_1)}$, then $Q^{(t)}$ will converge to a local minimum (or saddle point) of $\mathcal{F}_{bethe}$ with MN and Obs constraints satisfied.*

## 5   Experiments

In this section we report on two experiments on the feasibility of UPS. In the first experiment we compared the speed of convergence against other methods which minimize $\mathcal{F}_{bethe}$. In the second experiment we compared the accuracy of UPS against loopy IS. In both experiments we used $5 \times 5$ Boltzmann machines with states $\{0, 1\}$ and structure as shown in figure 1a. The weights are sampled randomly from a Gaussian with mean 0 and standard deviation $\sigma_w$ and the biases are sampled from a Gaussian with standard deviation $\sigma_b$ and mean $-\sum\{$ incoming weights $\}/2$. The means of the biases are shifted so that if $\sigma_b$ is small, the mean values of $x_i$ will be approximately $0.5$. The desired marginals are $\widehat{o}_i(1) = 1/(1 + e^{\alpha_i})$ where $\alpha_i$ are sampled from a Gaussian with mean 0 and standard deviation $\sigma_\alpha$.

**Experiment 1** *Speed of Convergence*

We compared the speed of convergence for the following algorithms: loopy IS, IS+BP, GIS+BP (parallel GIS with marginals estimated by loopy BP), UPS-H (clamping rows of nodes every iteration as in figure 1(b) and UPS-HV (alternatingly clamping rows and columns as in figures 1(b) and 1(c)). We tested the algorithms on 100 networks, with $\sigma_w = 5$, $\sigma_b = 1$ and $\sigma_\alpha = 4$. We find that the result is not sensitive to the settings of $\sigma_w, \sigma_b$ and $\sigma_\alpha$ so long as the algorithms are able to converge without damping. The result is shown in figure 1e. IS+BP and GIS+BP are slow because the loopy BP phase is expensive. UPS-H and UPS-HV both do better than IS+BP and GIS+BP because the inner loops are cheaper, and the Lagrange multipliers $\lambda_i(x_i)$ are updated more frequently. Further we see that UPS-HV is faster than UPS-H since information is propagated faster throughout the network. loopy IS is the fastest. However the next experiment shows that it also converges less frequently and there is a trade off between the speed of loopy IS and the stability of UPS.

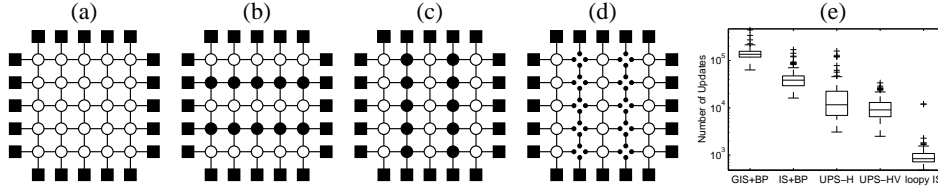

Figure 1: (a) Network structure. Circles are hidden nodes and black squares are observationally constrained nodes. (b) Clamping rows of nodes. Black circles are the clamped nodes. (c) Clamping columns of nodes. (d) Replicating each clamped and observed node in (c). (e) Speed of convergence. The box lines are at the median and upper and lower quartiles, and the whiskers describe the extent of data. An algorithm or subroutine is considered converged if the beliefs change by less than $10^{-10}$.

**Experiment 2** *Accuracy of Estimated Marginals*

We compared the accuracy of the posterior marginals obtained using UPS-HV and loopy IS for four possible types of constraints, as shown in figure 2. In case (a), the constraint marginals are delta functions, so that generalized inference reduces down to ordinary inference, loopy IS reduces to loopy BP and UPS becomes a convergent alternative to loopy BP. In case (b), we did not enforce any Obs constraints so that the problem is one of estimating the marginals of the prior $P(x)$. The general trend is that loopy BP and UPS are comparable, and they perform worse as weights get larger, biases get smaller or there is less evidence. This confirms the results in [6]. Further, we see that when loopy BP did not converge, UPS's estimates are not better than loopy BP's estimates. The reason this is happening is described in [6].

In cases (c) and (d) we set $\sigma_\alpha = 0.2, 2.0$, corresponding to $b_i(1) \approx 0.5$ and $b_i(1)$ spread out over $[0, 1]$ respectively. In these cases UPS and loopy IS did equally well when the latter converged, but UPS continued to perform well even when loopy IS did not converge. Since loopy BP always converged when UPS performed well (for cases (a) and (b)), and we used very high damping, we conclude that loopy IS's failure to converge must be due to performing scaling updates before accurate marginals were available. Concluding, we see that UPS is comparable to loopy IS when generalized inference reduces to ordinary inference, but in the presence of Obs constraints it is better.

## 6 Discussion

In this paper we have shown that approximating the KL divergence with the Bethe free energy leads to viable algorithms for approximate generalized inference. We also find that there is an interesting and fruitful relationship between IS and loopy BP. Our novel algorithm UPS can also be used as a convergent alternative to loopy BP for ordinary inference.

Interesting extensions are to cluster nodes together to get more accurate approximations to the KL divergence analogous to the Kikuchi free energy, and to handle marginal constraints over subsets of nodes. This will again lead to a close relationship between IS and junction tree propagation, but the details are to be worked out. We can also explore other algorithms to minimize $\mathcal{F}_{bethe}$, including the CCCP algorithm [7]. Another interesting direction for future work is algorithms for learning in log linear models by approximating the free energy.

## Footnotes

[1]To avoid confusion, we will explicitly use "ordinary inference" for normal inference, but when there is no confusion "inference" by itself will mean generalized inference. Ditto for posteriors.

[2]For a quick example, consider a two node Boltzmann machine, with weight 5 and biases $-2.5$, and the desired means on both nodes are 0.3. Then using either naive mean field or naive TAP equations to estimate the marginals required by IS will not converge.

[3] This was first shown in [2], with a different but equivalent identification of $M_j(x_j)$ and $\lambda_{ji}(x_i)$.

[4]For $i', j' \in G'$ define $\psi_{i'j'}(x_{i'}, x_{j'}) = \psi_{ij}(x_i, x_j)$ where $i$ and $j$ are the original nodes in $G$. Similarly for $\psi_{i'}(x_{i'})$, $b_{i'j'}(x_{i'}, x_{j'})$ and $b_{i'}(x_{i'})$.

## References

[1] J. Pearl. *Probabilistic reasoning in intelligent systems : networks of plausible inference*. Morgan Kaufmann Publishers, San Mateo CA, 1988.

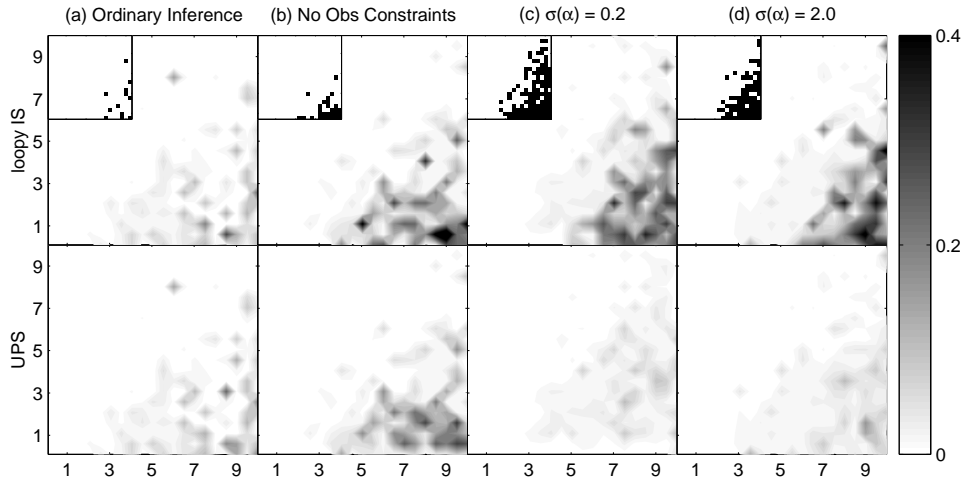

Figure 2: Each plot shows the mean absolute errors for various settings of $\sigma_w$ (x-axis) and $\sigma_b$ (y-axis). The top plots show errors for loopy IS and bottom plots show errors for UPS. The inset shows the cases (black) when loopy IS did not converge within 2000 iterations, with linear damping slowly increasing to 0.99.

[2] J.S. Yedidia, W. Freeman, and Y. Weiss. Generalized belief propagation. In *Advances in Neural Information Processing Systems*, volume 13, 2000.

[3] W. E. Deming and F. F. Stephan. On a least square adjustment of a sampled frequency table when the expected marginal totals are known. *Annals of Mathematical Statistics*, 11:427–444, 1940.

[4] J. Darroch and D. Ratcliff. Generalized iterative scaling for log-linear models. *Annals of Mathematical Statistics*, 43:1470–1480, 1972.

[5] K. Murphy, Y. Weiss, and M. Jordan. Loopy belief propagation for approximate inference : An empirical study. In *Proceedings of the Conference on Uncertainty in Artificial Intelligence*, volume 15. Morgan Kaufmann Publishers, 1999.

[6] M. Welling and Y. W. Teh. Belief optimization for binary networks : A stable alternative to loopy belief propagation. In *Uncertainty in Artificial Intelligence*, 2001.

[7] A. L. Yuille. CCCP algorithms to minimize the Bethe and Kikuchi free energies: Convergent alternatives to belief propagation. 2002.
